# Effective Split-Merge Monte Carlo Methods for Nonparametric Models of Sequential Data

**Michael C. Hughes**[1], **Emily B. Fox**[2], and **Erik B. Sudderth**[1]

[1]Department of Computer Science, Brown University, {`mhughes,sudderth`}`@cs.brown.edu`
[2]Department of Statistics, University of Washington, `ebfox@stat.washington.edu`

## Abstract

Applications of Bayesian nonparametric methods require learning and inference algorithms which efficiently explore models of unbounded complexity. We develop new Markov chain Monte Carlo methods for the beta process hidden Markov model (BP-HMM), enabling discovery of shared activity patterns in large video and motion capture databases. By introducing split-merge moves based on sequential allocation, we allow large global changes in the shared feature structure. We also develop data-driven reversible jump moves which more reliably discover rare or unique behaviors. Our proposals apply to any choice of conjugate likelihood for observed data, and we show success with multinomial, Gaussian, and autoregressive emission models. Together, these innovations allow tractable analysis of hundreds of time series, where previous inference required clever initialization and lengthy burn-in periods for just six sequences.

## 1 Introduction

Bayesian nonparametric time series models, including various "infinite" Markov switching processes [1, 2, 3], provide a promising modeling framework for complex sequential data. We focus on the problem of discovering coherent, short-term activity patterns, or "behaviors", shared among related time series. For example, given collections of videos or human motion capture sequences, our goal is to (i) identify a concise global library of behaviors that explain the observed motions, (ii) annotate each sequence with the subset of behaviors used, and (iii) label each timestep with one active behavior. The *beta process hidden Markov model* (BP-HMM) [4] offers a promising solution to such problems, allowing an unbounded set of relevant behaviors to be learned from data.

Learning BP-HMMs from large datasets poses significant computational challenges. Fox et al. [4] considered a dataset containing only six motion capture sequences and proposed a Markov chain Monte Carlo (MCMC) method that required careful initialization and tens of thousands of burn-in iterations. Their sampler included innovations like block state sequence resampling [5] and marginalization of some variables. However, like most MCMC samplers, their proposals only modified small subsets of variables at each step. Additionally, the sampler relied on parameter proposals from priors, leading to low acceptance rates for high-dimensional data. Alternative single-site MCMC moves, such as those based on slice sampling [6, 7], can exhibit similarly slow mixing. Our goal is to expose this pervasive issue with conventional MCMC, and develop new samplers that rapidly explore the structural uncertainty inherent in Bayesian nonparametric models. While our focus is on the BP-HMM, the technical innovations underlying our samplers are much more broadly applicable.

We make two complementary improvements to previous BP-HMM samplers [4]. First, we develop split-merge moves which change many variables simultaneously, allowing rapid improvements in the discovered behavior library. Our approach builds on previous work on restricted Gibbs proposals [8] and sequential allocation strategies [9], both of which were formulated for static Dirichlet

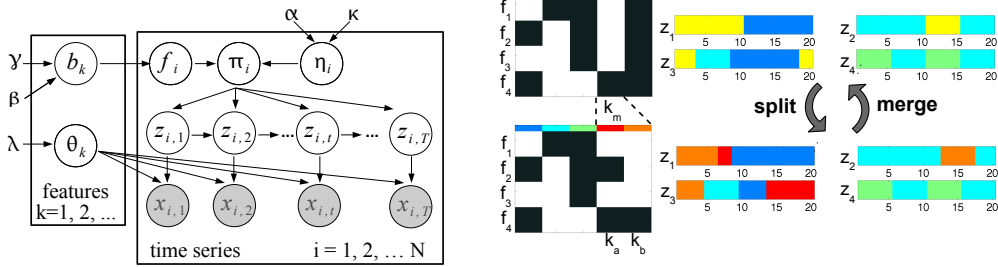

Figure 1: *Left:* The BP-HMM as a directed graphical model. *Right:* Illustration of our split and merge proposals, which modify both binary feature assignment matrices $\mathbf{F}$ (white indicates present feature) and state sequences $\mathbf{z}$. We show $\mathbf{F}$, $\mathbf{z}$ before (*top*) and after (*bottom*) feature $k_m$ (yellow) is split into $k_a, k_b$ (red,orange). An item $i$ with $f_{ik_m} = 1$ can have either $k_a, k_b$, or both after the split, and its new $\mathbf{z}_i$ sequence can use any features available in $\mathbf{f}_i$. An item without $k_m$ cannot possess $k_a, k_b$, and its state sequence $\mathbf{z}_i$ does not change.

process (DP) mixture models [10]. Second, we design *data-driven* [11] reversible jump moves [12] which efficiently discover behaviors unique to a single sequence. These data-driven proposals are especially important for high-dimensional observation sequences. Both innovations apply to any likelihood model with a conjugate prior; we show success with multinomial models of discrete video descriptors, and Gaussian autoregressive models of continuous motion capture trajectories.

We begin in Sec. 2 by reviewing the BP-HMM model. We describe previous BP-HMM samplers [4] in Sec. 3.1, and then develop our novel split-merge and data-driven proposals in Sec. 3.3-3.4. We evaluate our contributions in Sec. 4, showing improvements over prior work modeling human motions captured both via a marker-based mocap system [4] and video cameras [13].

## 2 Beta Process Hidden Markov Models

Latent feature models intuitively capture the *sparse sharing patterns* occurring in collections of human action sequences. For example, one sequence may contain jogging and boxing, while another has jogging and dancing. We assign the $i^{th}$ sequence or "item" with a sparse binary vector $\mathbf{f}_i = [f_{i1}, f_{i2}, \ldots]$ indicating the presence or absence of each feature in the unbounded global collection. Given $N$ items, corpus-wide assignments are denoted by a binary matrix $\mathbf{F}$ whose $i^{th}$ row is $\mathbf{f}_i$.[1]

The feature matrix $\mathbf{F}$ is generated by an underlying stochastic process, the *beta process* (BP) [14]:

$$B \mid B_0, \gamma, \beta \sim \text{BP}(\beta, \gamma B_0), \quad B = \sum_{k=1}^{\infty} b_k \delta_{\theta_k}. \tag{1}$$

A realization $B$ of the BP contains infinitely many features $k$. For each feature, $\theta_k \sim B_0$ marks its data-generation parameters, while $b_k \in (0, 1)$ denotes its inclusion probability. The binary feature vector for item $i$ is determined by independent Bernoulli draws $f_{ik} \sim \text{Ber}(b_k)$. Marginalizing over $B$, the number of active features in item $i$ has distribution $\text{Poisson}(\gamma)$, determined by *mass parameter* $\gamma$. The *concentration parameter* $\beta$ controls how often features are shared between items [15].

To apply feature models to time series data, Fox et al. [4] combine the BP with a hidden Markov model to form the BP-HMM, shown in Fig. 1. The binary vector $\mathbf{f}_i$ determines a finite set of states available for item $i$. Each timestep $t$ is assigned a single state $z_{it} = k$ from the set $\{k : f_{ik} = 1\}$, determining which parameters $\theta_k$ generate data $x_{it}$. Many different data-generation models are possible. As in [4], for motion capture data we use a first-order Gaussian autoregressive process with parameters $\theta_k = (A_k, \Sigma_k)$ drawn from a matrix-normal inverse-Wishart conjugate prior

$$x_{it} \mid z_{it} = k, x_{it-1} \sim \mathcal{N}(A_k x_{it-1}, \Sigma_k) \qquad A_k, \Sigma_k \mid B_0 \sim \mathcal{MNW}^{-1}(\nu, S_0, R_0) \tag{2}$$

To study video, we use a Dirichlet-multinomial model for quantized interest point descriptors [13]

$$x_{it} \mid z_{it} = k \sim \text{Multinomial}(\theta_k) \qquad \theta_k \mid B_0 \sim \text{Dirichlet}(\lambda_0, \lambda_0, \ldots \lambda_0) \tag{3}$$

The BP-HMM allows each item *independent* transition dynamics. The transition distribution $\pi_{ij}$ from each state $j$ for the HMM of item $i$ is built by drawing a set of transition weights $\eta_i$, and then normalizing these over the set of active features $\mathbf{f}_i$:

$$\eta_{ijk} \sim \text{Gamma}(\alpha + \kappa\delta_{jk}, 1), \qquad \pi_{ijk} = \frac{\eta_{ijk}f_{ik}}{\sum_\ell f_{i\ell}\eta_{ij\ell}}. \qquad (4)$$

Here, $\delta_{jk} = 1$ if $j = k$, and 0 otherwise. This construction assigns positive transition mass $\pi_{ijk}$ only to features $k$ active in $\mathbf{f}_i$. The *sticky* parameter $\kappa$ places extra expected mass on self-transitions [3], biasing the model to learn state sequences $\mathbf{z}$ with temporally persistent states.

## 3 MCMC Inference with Split-Merge Proposals

We first summarize the MCMC methods previously proposed for the BP-HMM [4]. We then present our novel contributions: split-merge moves and data-driven reversible jump proposals. Full algorithmic details for all samplers are found in the supplement, and our code is available online.

### 3.1 Local Monte Carlo Proposals

Fox et al. [4]'s sampler alternates updates to HMM parameters $\theta$ and $\eta$, discrete state sequences $\mathbf{z}$, and feature assignments $\mathbf{F}$. Fixing $\mathbf{F}$ defines a collection of finite HMMs, so each $\mathbf{z}_i$ can be block sampled by dynamic programming [5], and then $\theta, \eta$ drawn from conjugate posteriors.[2] Sampling each item's features requires separate updates to features *shared* with some other time series and features *unique* to item $i$. Both updates marginalize state sequences $\mathbf{z}$ and inclusion weights $\mathbf{b}$.

For each *shared* feature $k$, Fox et al. propose flipping $f_{ik}$ to the complementary binary value and accept or reject according to the Metropolis-Hastings (MH) rule. This local move alters only one entry in $\mathbf{F}$ while holding all others fixed; the split-merge moves of Sec. 3.3 improve upon it.

For *unique* features, Fox et al. [4] define a reversible pair of birth and death moves which add or delete features to single sequences. While this approach elegantly avoids approximating the infinite BP-HMM, their birth proposals use the (typically vague) prior to propose emission parameters $\theta_{k^*}$ for new features $k^*$. We remedy this slow exploration with data-driven proposals in Sec. 3.4.

### 3.2 Split-Merge Proposals for Dirichlet Process Models

Split-merge MCMC methods were first applied to nonparametric models by Jain and Neal [8] in work focusing on DP mixture models with conjugate likelihoods. Conjugacy allows samplers to operate directly on discrete partitions of observations into clusters, marginalizing emission parameters. Jain and Neal present valid proposals that reversibly split a single cluster $k_m$ into two $(k_a, k_b)$, or merge two clusters into one. Since merges are deterministic, the primary contribution of [8] is a generic technique – *restricted Gibbs* (RG) sampling – for proposing splits consistent with the data.

To construct an initial split of $k_m$, the RG sampler first assigns items in cluster $k_m$ at random to either $k_a$ or $k_b$. Starting from this partition, the proposal is constructed by performing one-at-a-time Gibbs updates, forgetting an item's current cluster and reassigning to either $k_a$ or $k_b$ conditioned on the remaining partitioned data. After several sweeps, these Gibbs updates encourage proposed clusters $k_a$ and $k_b$ which agree with the data and thus are more likely to be accepted. For non-conjugate models, more complex RG proposals can be constructed which instantiate emission parameters [16].

Even in small datasets, there can be significant benefits from performing five or more sweeps for each RG proposal [8]. For large datasets, however, requiring many sweeps for a single proposal is computationally expensive. An alternative *sequential allocation* [9] method replaces the random initialization of RG by using two randomly chosen items to "anchor" the two new clusters $k_a, k_b$. Remaining items are then *sequentially* assigned to either $k_a$ or $k_b$ one-at-a-time, using RG moves conditioning only on previously assigned data. This creates a proposed partition in agreement with the data after only one sampling sweep. Recent work has shown some success with sequentially-allocated split-merge moves for a hierarchical DP topic model [17].

For nonparametric models not based on the DP, split-merge moves are not well studied. Several authors have considered RG split-merge proposals for beta process models [18, 19, 20]. However, these papers lack technical details, and do not contain experiments showing improved mixing.

### 3.3 Split-Merge Proposals for the BP-HMM

We now adapt RG and sequential allocation to define BP-HMM split-merge moves. In the mixture models considered by prior work [8, 9], each data item $i$ is associated with a single cluster $k_i$, so selecting two anchors $i$, $j$ also identifies two cluster indices $k_i, k_j$. However, in feature-based models such as the BP-HMM, each data item $i$ is associated with a *collection* of features indexed by $\mathbf{f}_i$. Therefore, our proposals require mechanisms for selecting anchors *and* for choosing candidate states to split or merge from $\mathbf{f}_i, \mathbf{f}_j$. Additionally, our proposals must allow changes to state sequences $\mathbf{z}$ to reflect changes in $\mathbf{F}$. Our proposals thus jointly create a new configuration $(\mathbf{F}^*, \mathbf{z}^*)$, collapsing away HMM parameters $\theta, \eta$. Fig. 1 illustrates $(\mathbf{F}, \mathbf{z})$ before and after a split move.

**Selecting Anchors** Following [9], we first randomly select distinct anchor data items $i$ and $j$. The fixed choice of $i, j$ defines a split-merge transition kernel satisfying detailed balance. Next, we select from each anchor one feature it possesses, denoted $k_i, k_j$. This choice determines the proposed move: we merge $k_i, k_j$ if they are distinct, and split $k_i = k_j$ into two new features otherwise.

Selecting $k_i, k_j$ uniformly at random is problematic. First, in datasets with many features choosing $k_i = k_j$ is unlikely, making split moves rare. We need to bias the selection process to propose splits more often. Second, in a well-trained model most feature pairs will not make a sensible merge. Selecting a pair that explains similar data is crucial for efficiency. We thus develop a proposal distribution which first draws $k_i$ uniformly from $\mathbf{f}_i$, and then selects $k_j$ given fixed $k_i$ as follows:

$$q_k(k_i, k_j) = \text{Unif}(k_i \mid \mathbf{f}_i)q(k_j \mid k_i, \mathbf{f}_j), \quad q(k_j = k \mid k_i, \mathbf{f}_j) \propto \begin{cases} 2C_j f_{jk} & \text{if } k = k_i \\ f_{jk}\frac{m(\mathbf{x}_{k_i}, \mathbf{x}_k)}{m(\mathbf{x}_{k_i})m(\mathbf{x}_k)} & \text{o.w.} \end{cases} \quad (5)$$

Here, $\mathbf{x}_k$ is the data assigned to $k$, $m(\cdot)$ denotes the *marginal likelihood* of data collapsing away emission parameters $\theta$, and $C_j = \sum_{k_j \neq k_i} f_{jk_j} m(\mathbf{x}_{k_i}, \mathbf{x}_{k_j})/ \left[m(\mathbf{x}_{k_i})m(\mathbf{x}_{k_j})\right]$. This construction gives large mass (2/3) to a split move when possible, and also encourages choices $k_i \neq k_j$ for a merge that explain similar data via the marginal likelihood ratio. A large ratio means the model prefers to explain all data assigned to $k_i, k_j$ *together* rather than separately, biasing selection towards promising merge candidates. We find higher acceptance rates for merges under this $q_k$, which justifies the small cost of computing $m(\cdot)$ from cached sufficient statistics.

Once $k_i, k_j$ are fixed, we construct the candidate state $\mathbf{F}^*, \mathbf{z}^*$. As shown in Fig. 1, we only alter $\mathbf{f}_\ell, \mathbf{z}_\ell$ for items $\ell$ which possess either $k_i$ or $k_j$. We call this set of items the *active set* $\mathcal{S}$.

**Split** Our split proposal is defined in Alg. 1. Sweeping through a random permutation of items $\ell$ in the active set $\mathcal{S}$, we draw each item's assignment to new features $k_a, k_b$ and resample its state sequence. We sample $[f^*_{\ell k_a} f^*_{\ell k_b}]$ from its conditional posterior given previously visited items in $\mathcal{S}$, requiring that $\ell$ must possess at least one of the new features. We then block sample its state sequence $\mathbf{z}_\ell^*$ given $\mathbf{f}_\ell^*$. The dynamic programming recursions underlying these proposals use non-random auxiliary parameters: $\hat{\eta}_\ell$ is set to its prior mean, and $\hat{\theta}_k$ to its posterior mean given the current $\mathbf{z}$. For new states $k^* \in \{k_a, k_b\}$, we initialize $\hat{\theta}_{k^*}$ from anchor sequences and then update to account for new data assigned to $k^*$ at each item $\ell$. This enables better matching of proposed features to data statistics. Finally, we sample $\mathbf{f}^*, \mathbf{z}^*$ for anchor items, enforcing $f^*_{ik_a} = 1$ and $f^*_{jk_b} = 1$ so the move remains reversible under a merge. This does not force $\mathbf{z}_i^*$ to use $k_a$ nor $\mathbf{z}_j^*$ to use $k_b$.

**Merge** For a merge move, constructing $\mathbf{F}^*$ is deterministic: we set $f^*_{\ell k_m} = 1$ for $\ell \in \mathcal{S}$, and 0 otherwise. We thus need only to sample $\mathbf{z}_\ell^*$ for items in $\mathcal{S}$, using a block sampler as in Alg. 1. Again this requires auxiliary HMM parameters $\hat{\theta}, \hat{\eta}$, which we emphasize are deterministic tools enabling *collapsed* proposals of discrete indicators $\mathbf{F}^*, \mathbf{z}^*$. We never sample $\theta, \eta$.

**Accept-Reject** After drawing a candidate value $\mathbf{F}^*, \mathbf{z}^*$, the final step is to compute a Metropolis-Hastings acceptance probability $\min(\rho, 1)$. We give the ratio for a *split* move which creates features

**Alg. 1** Propose *split* of feature $k_m$ into $k_a, k_b$ given $\mathbf{F}, \mathbf{z}, \mathbf{x}$, anchor items $i, j$, set $\mathcal{S}=\{\ell{:}f_{\ell,k_m}{=}1\}$

| | | | |
|---|---|---|---|
| 1: | $f_{i,[k_a,k_b]} \leftarrow \begin{bmatrix}1 & 0\end{bmatrix}$ | $z_{i,t:z_{i,t}=k_m} \leftarrow k_a$ | *use anchor $i$ to create $k_a$* |
| 2: | $f_{j,[k_a,k_b]} \leftarrow \begin{bmatrix}0 & 1\end{bmatrix}$ | $z_{j,t:z_{j,t}=k_m} \leftarrow k_b$ | *use anchor $j$ to create $k_b$* |
| 3: | $\hat{\theta} \leftarrow \mathbb{E}\left[\theta \mid \mathbf{x}, \mathbf{z}, \lambda\right]$ | $\hat{\eta}_\ell \leftarrow \mathbb{E}\left[\eta_\ell \mid \alpha, \kappa\right], \ell \in \mathcal{S}$ | *set HMM params deterministically* |
| 4: | $\mathcal{S}_{\text{prev}} = \{i, j\}$ | | *initialize set of previously visited items* |

5: **for** non-anchor items $\ell$ in random permutation of active set $\mathcal{S}$:

6: $\qquad f_{\ell,[k_a k_b]} \sim \begin{cases} \begin{bmatrix}0 & 1\end{bmatrix} \\ \begin{bmatrix}1 & 0\end{bmatrix} \propto p(f_{\ell,[k_a k_b]} \mid f_{\mathcal{S}_{\text{prev}},[k_a k_b]})p(\mathbf{x}_\ell \mid \mathbf{f}_\ell, \hat{\theta}, \hat{\eta}_\ell) \\ \begin{bmatrix}1 & 1\end{bmatrix} \end{cases}$

7: $\qquad \mathbf{z}_\ell \sim p(\mathbf{z}_\ell \mid \mathbf{x}_\ell, \mathbf{f}_\ell, \hat{\theta}, \hat{\eta}_\ell)$  *draw $f, z$ and update $\hat{\theta}$ for each item*

8: $\qquad$ add $\ell$ to $\mathcal{S}_{\text{prev}}$  *condition on previously visited items*

9: $\qquad$ **for** $k = k_a, k_b : \quad \hat{\theta}_k \leftarrow \mathbb{E}\left[\theta_k \mid \lambda, \{x_{nt} : z_{nt} = k, n \in \mathcal{S}_{\text{prev}}\}\right]$

10: $f_{i,[k_a k_b]} \sim \begin{cases}\begin{bmatrix}1 & 0\end{bmatrix} \\ \begin{bmatrix}1 & 1\end{bmatrix}\end{cases} \qquad f_{j,[k_a k_b]} \sim \begin{cases}\begin{bmatrix}0 & 1\end{bmatrix} \\ \begin{bmatrix}1 & 1\end{bmatrix}\end{cases}$  *finish by sampling $f, z$ for anchors*

11: $\mathbf{z}_i \sim p(\mathbf{z}_i \mid \mathbf{x}_i, \mathbf{f}_i, \hat{\theta}, \hat{\eta}_i) \qquad \mathbf{z}_j \sim p(\mathbf{z}_j \mid \mathbf{x}_j, \mathbf{f}_j, \hat{\theta}, \hat{\eta}_j)$

$k_a, k_b$ from $k_m$ below. The acceptance ratio for a merge move is the reciprocal of Eq. (6).

$$\rho_{\text{split}} = \frac{p(\mathbf{x}, \mathbf{z}^*, \mathbf{F}^*)}{p(\mathbf{x}, \mathbf{z}, \mathbf{F})} \frac{q_{\text{merge}}(\mathbf{F}, \mathbf{z} \mid \mathbf{x}, \mathbf{F}^*, \mathbf{z}^*, k_a, k_b)}{q_{\text{split}}(\mathbf{F}^*, \mathbf{z}^* \mid \mathbf{x}, \mathbf{F}, \mathbf{z}, k_m)} \frac{q_k(k_a, k_b \mid \mathbf{x}, \mathbf{F}^*, \mathbf{z}^*, i, j)}{q_k(k_m, k_m \mid \mathbf{x}, \mathbf{F}, \mathbf{z}, i, j)} \quad (6)$$

The joint probability $p(\mathbf{x}, \mathbf{z}, \mathbf{F})$ is only tractable with conjugate likelihoods. Proposals which instantiate emission parameters $\theta$, as in [16], would be required in the non-conjugate case.

### 3.4 Data-Driven Reversible Jump Birth and Death Proposals

Efficiently adding or deleting unique features is crucial for good mixing. To accept the birth of new feature $k^* = K + 1$ for item $i$, this feature must explain some of the observed data $\mathbf{x}_i$ at least as well as existing features $1, 2, \ldots K$. High-dimensional emission parameters $\theta_{k^*}$ drawn from a vague prior are unlikely to match the data at hand. Instead, we suggest a *data-driven* proposal [11, 13] for $\theta_{k^*}$. First, select at random a subwindow $W$ of the current sequence $i$. Next, use data in this subwindow $\mathbf{x}_W = \{x_{it} : t \in W\}$ to create a proposal distribution: $q_\theta(\theta) = \frac{1}{2}p(\theta) + \frac{1}{2}p(\theta \mid \mathbf{x}_W)$, which is a mixture of $\theta$'s prior and posterior given $\mathbf{x}_W$. This mixture strikes a balance between proposing promising new features (via the posterior) while also making death moves possible, since the diffuse prior will place some mass on the reverse birth move.

Let $U_i$ denote the number of unique features in $\mathbf{f}_i$ and $\nu = \gamma \frac{\beta}{N-1+\beta}$. The acceptance probability for a birth move to candidate state $\mathbf{f}_i^*, \eta_i^*, \theta^*$ is then $\min(\rho_{\text{birth}}, 1)$, where

$$\rho_{\text{birth}} = \frac{p(\mathbf{x}_i \mid \mathbf{f}_i^*, \eta_i^*, \theta^*)}{p(\mathbf{x}_i \mid \mathbf{f}_i, \eta_i, \theta)} \frac{\text{Poi}(U_i + 1 \mid \nu)}{\text{Poi}(U_i \mid \nu)} \frac{p_\theta(\theta_{k^*}^*)}{q_\theta(\theta_{k^*}^*)} \frac{q_f(\mathbf{f}_i \mid \mathbf{f}_i^*)}{q_f(\mathbf{f}_i^* \mid \mathbf{f}_i)} \quad (7)$$

Eq. (7) is similar to the ratio for the birth proposal from the prior, adding only one term to account for the proposed $\theta_{k^*}^*$. Note that each choice of $W$ defines a valid pair of birth-death moves satisfying detailed balance, so we need not account for this choice in the acceptance ratio [21].

## 4 Experimental Results

Our experiments compare competing inference algorithms for the BP-HMM; for comparisons to alternative models, see [4]. To evaluate how well our novel moves explore the posterior distribution, we compare three possible methods for adding or removing features: split-merge moves (SM, Sec. 3.3), data-driven moves (DD, Sec. 3.4), and reversible jump moves using the prior (Prior [4], Sec. 3.1). All experiments interleave the chosen update with the standard MH updates to shared features of $\mathbf{F}$ and Gibbs updates to HMM parameters $\theta, \eta$ described in Sec. 3.1.

For each comparison, we run multiple initializations and rank the chains from "best" to "worst" according to joint probability. Each chain is allowed the same amount of computer time.

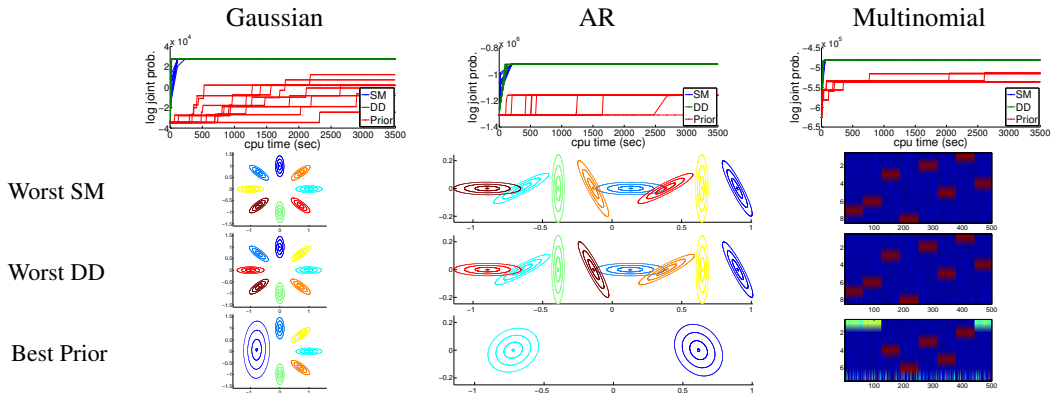

Figure 2: Feature creation for synthetic data with Gaussian (*left*), AR (*middle*), or multinomial (*right*) likelihoods. Each run begins with one feature used by all items, and must add new features via split-merge proposals (SM), or reversible-jump moves using data-driven (DD) or prior (Prior) proposals. *Top:* Log joint probability versus computation time, for 10 random initializations of each sampling algorithm. *Bottom:* Emission parameters associated with the last sample after one hour of computation time. Gaussian $\theta = (\mu, \Sigma)$ and AR $\theta = (A, \Sigma)$ shown as contour lines in first two dimensions, with location determined by $\mu, A$. Multinomial $\theta$ shown as image where row $k$ gives the emission distribution over vocabulary symbols for state $k$.

## 4.1 Synthetic Data

We examined toy datasets generated by a known set of 8 features (behaviors) $\theta_{\text{true}}$. To validate that our contributions apply for many choices of likelihood, we create three datasets: multinomial "bag-of-words" emissions using 500 vocabulary words, 8–dimensional Gaussian emissions, and a first-order autoregressive (AR) processes with 5 dimensions. Each dataset has $N = 100$ sequences.

First, we study how well each method *creates* necessary features from scratch. We initialize the sampler with just one feature used by all items, and examine how many true states are recovered after one hour of computation time across 10 runs. We show trace plots as well as illustrations of recovered emission parameters $\theta$ in Fig. 2. All runs of both SM and DD moves find all true states within several minutes, while no Prior run recovers all true states, remaining stuck with merged versions of true features. DD moves add new features most rapidly due to low computational cost.

We next examine whether each inference method can *remove* unnecessary features. We consider a different toy dataset of several hundred sequences and a redundant initialization in which 2 copies of each true state exist. Half of the sequences are initialized with $\mathbf{f}, \mathbf{z}$ set to corresponding true values in copy 1, and the second half using copy 2. Using Gaussian and AR likelihoods, all SM runs merge down to the 8 true states, at best within five minutes, but no DD or Prior run ever reaches this optimal configuration in the allotted hour. Merge moves enable critical global changes, while the one-at-a-time updates of [4] (and our DD variant) must take long random walks to completely delete a popular feature. Further details are provided in the supplementary material.

These results demonstrate the importance of DD birth and split moves for exploring new features, and merge moves for removing features via proposals of large assignment changes. As such, we consider a sampler that interleaves SM and DD moves in our subsequent analyses.

## 4.2 Motion Capture Data

We next apply our improved MCMC methods to motion capture (mocap) sequences from the CMU database [22]. First, we consider the small dataset examined by [4]: 6 sequences of physical exercises with motion measurements at 12 joint angles, modeled with an AR(1) likelihood. Human annotation identifies 12 actions, which we take as ground truth. Previous results [4] show that the BP-HMM outperforms competitors in segmenting these actions, although they report that some true actions like jogging are split across multiple recovered features (see their Fig. 5). We set likelihood hyperparameters similarly to [4], with further details provided in the supplement.

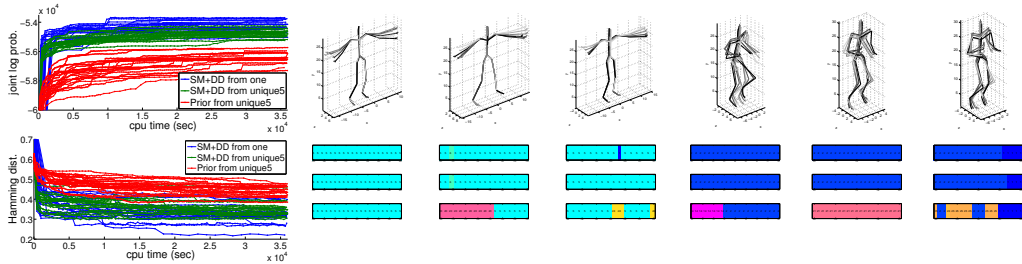

Figure 3: Analysis of 6 motion capture sequences previously considered by [4]. *Left:* Joint log probability and Hamming distance (from manually annotated behaviors) for 20 runs of each method over 10 hours. *Right:* Examples of arm circles and jogging from 3 sequences, along with estimated $z_i$ of last sample from the best run of each method. SM+DD moves (*top row* started from `one` feature, *middle row* started with 5 `unique` states per sequence) successfully explain each action with one primary state, while [4]'s sampler (*bottom row*) started from 5 `unique` features remains stuck with multiple unique states for one true action.

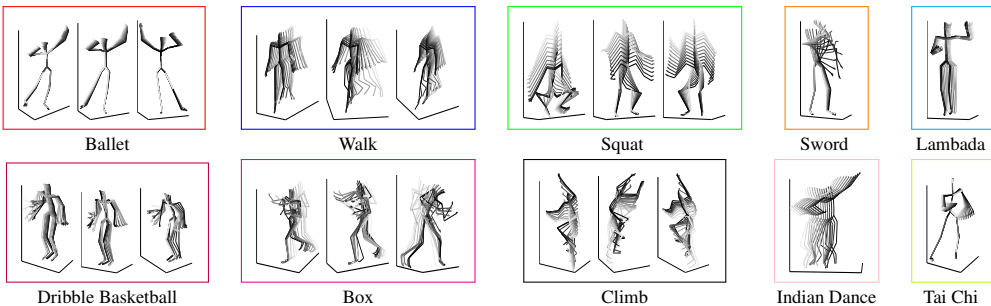

Figure 4: Analysis of 124 mocap sequences, showing 10 of 33 recovered behaviors. Skeleton trajectories are built from contiguous segments $\geq 1$ sec long assigned to each behavior. Boxed groups contain segments from distinct sequences assigned to the same state. Some states only appear in one sequence.

In Fig. 3, we compare a sampler which interleaves SM and DD moves with [4]'s original method. We run 20 chains of each method for ten hours from two initializations: `unique5`, which assigns 5 unique features per sequence (as done in [4]), and `one`, using a single feature across all items. In both log probability and Hamming distance, SM+DD methods are noticeably superior. Most interestingly, SM+DD starting from a parsimonious `one` feature achieves best performance overall, showing that clever initialization is not necessary with our algorithm. The best run of SM+DD from `one` achieves Hamming distance of 0.22, compared to 0.3 reported in [4]. No Prior proposal run from `one` created any additional states, indicating the importance of using our improved methods of feature exploration even in moderate dimensions.

Our SM moves are critical for effectively creating and deleting features. Example segmentations of arm-circles and jogging actions in Fig. 3 show that SM+DD consistently use one dominant behavior across all segments where the action appears. In contrast, the Prior remains stuck with some unique behaviors used in different sequences, yielding lower probability and larger Hamming distance.

Next, we study a larger dataset of 124 sequences, all "Physical Activities & Sports" examples from the CMU mocap dataset. Analyzing a dataset of this size is computationally infeasible using the methods of [4]. Initializing from `unique5` would create over 500 features, requiring a prohibitively long sampling run to merge related behaviors. When initialized from `one`, the Prior sampler creates no additional features. In contrast, starting from `one`, our SM+DD moves rapidly identify a diverse set of 33 behaviors. A set of 10 behaviors representative of this dataset are shown in Fig. 4. Our improved algorithm robustly explores the posterior space, enabling this large-scale analysis.

### 4.3 CMU Kitchen: Activity Discovery from Video

Finally, we apply our new inference methods to discover common motion patterns from videos of recipe preparation in the CMU Kitchen dataset [23]. Each video is several minutes long and depicts a single actor in the same kitchen preparing either a pizza, a sandwich, a salad, brownies, or eggs. Our previous work [13] showed promising results in activity discovery with the BP-HMM using a

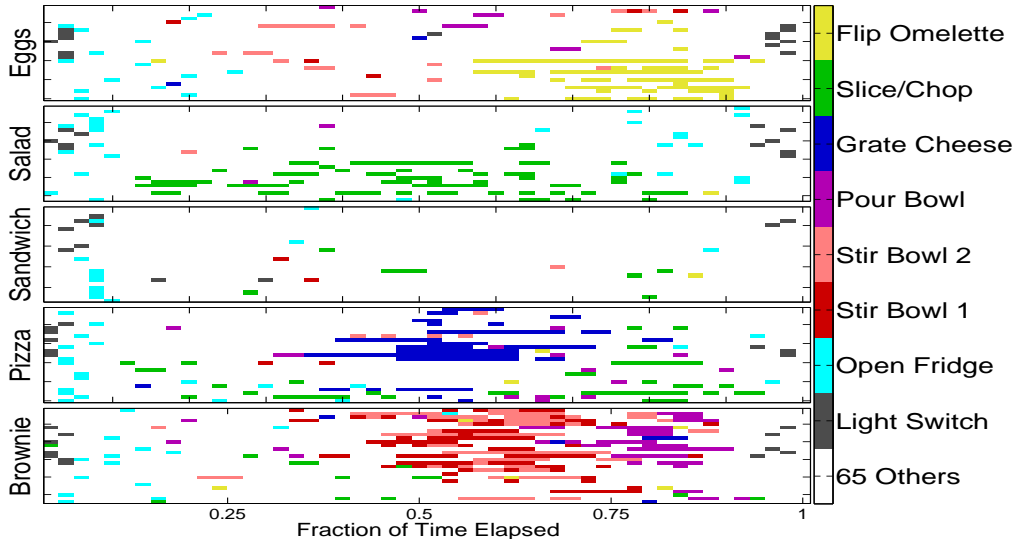

Figure 5: Activity discovery with 126 Kitchen videos, showing locations of select behaviors over time. Each row summarizes $z_i$ for a single video, labeled at left by recipe type (label not provided to the BP-HMM). We show only behaviors assigned to at least two timesteps in a local window.

small collection of 30 videos from this collection. We compare our new SM moves on this small dataset, and then study a larger set of 126 Kitchen videos using our improved sampler.

Using only the 30 video subset, Fig. 6 compares the combined SM+DD sampler with just DD or Prior moves, using fixed hyperparameter settings as in [13] and starting with just `one` feature. DD proposals offer significant gains over the prior, and further interleaving DD and SM moves achieves the best overall configuration, showing the benefits of proposing global changes to $\mathbf{F}, \mathbf{z}$.

Finally, we run the SM+DD sampler on 126 Kitchen sequences, choosing the best of 4 chains after several days of computer time (trace plots show convergence in half this time). Fig. 5 maps behavior assignments over time across all five recipes, using the last MCMC sample. Several intuitive behavior sharing patterns exist: chopping happens with carrots (salad) and pepperoni (pizza), while stirring occurs when preparing brownies and eggs. Non-uniform behavior usage patterns within a category are due to differences in available cooking equipment across videos. Please see the supplement for more experimental details and results.

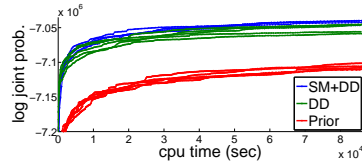

Figure 6: Joint log probability versus computation time for various samplers on the CMU Kitchen data [23] previously studied by [13].

# 5 Discussion

We have developed efficient MCMC inference methods for the BP-HMM. Our proposals do not require careful initialization or parameter tuning, and enable exploration of large datasets intractable under previous methods. Our approach makes efficient use of data and applies to any choice of conjugate emission model. We expect the guiding principles of data-driven and sequentially-allocated proposals to apply to many other models, enabling new applications of nonparametric analysis.

# Acknowledgments

M. Hughes was supported in part by an NSF Graduate Research Fellowship under Grant No. DGE0228243. E. Fox was funded in part by AFOSR Grant FA9550-12-1-0453.

## Footnotes

[1]Throughout this paper, for variables $w_{ijk}$ we use $\mathbf{w}$ to denote the vector or collection of $w_{ijk}$'s over the entire set of subscripts, and $\mathbf{w}_i$ for the collection over only the omitted subscripts $j$ and $k$

[2]Fox et al. [4] contains a small error in the resampling of $\eta$, as detailed and corrected in the supplement.

# References

[1] M. Beal, Z. Ghahramani, and C. Rasmussen. The infinite hidden Markov model. In *NIPS*, 2002.

[2] Y. W. Teh, M. I. Jordan, M. J. Beal, and D. M. Blei. Hierarchical Dirichlet processes. *Journal of the American Statistical Association*, 101(476):1566–1581, 2006.

[3] E. B. Fox, E. B. Sudderth, M. I. Jordan, and A. S. Willsky. A sticky HDP-HMM with application to speaker diarization. *Annals of Applied Statistics*, 5(2A):1020–1056, 2011.

[4] E. B. Fox, E. B. Sudderth, M. I. Jordan, and A. S. Willsky. Sharing features among dynamical systems with beta processes. In *NIPS*, 2010.

[5] S. L. Scott. Bayesian methods for hidden Markov models: Recursive computing in the 21st century. *JASA*, 97(457):337–351, 2002.

[6] J. Van Gael, Y. Saatci, Y. W. Teh, and Z. Ghahramani. Beam sampling for the infinite hidden Markov model. In *ICML*, 2008.

[7] C. Yau, O. Papaspiliopoulos, G. O. Roberts, and C. Holmes. Bayesian non-parametric hidden Markov models with applications in genomics. *JRSS B*, 73(1):37–57, 2011.

[8] S. Jain and R.M. Neal. A split-merge Markov chain Monte Carlo procedure for the Dirichlet process mixture model. *Journal of Computational and Graphical Statistics*, 13(1):158–182, 2004.

[9] D. B. Dahl. Sequentially-allocated merge-split sampler for conjugate and nonconjugate Dirichlet process mixture models. *Submitted to Journal of Computational and Graphical Statistics*, 2005.

[10] M. D. Escobar and M. West. Bayesian density estimation and inference using mixtures. *JASA*, 90(430):577–588, 1995.

[11] Z. Tu and S. C. Zhu. Image segmentation by data-driven Markov chain Monte Carlo. *PAMI*, 24(5):657–673, 2002.

[12] P.J. Green. Reversible jump Markov chain Monte Carlo computation and Bayesian model determination. *Biometrika*, 82(4):711–732, 1995.

[13] M. C. Hughes and E. B. Sudderth. Nonparametric discovery of activity patterns from video collections. In *CVPR Workshop on Perceptual Organization in Computer Vision*, 2012.

[14] R. Thibaux and M. I. Jordan. Hierarchical beta processes and the Indian buffet process. In *AISTATS*, 2007.

[15] T. L. Griffiths and Z. Ghahramani. Infinite latent feature models and the Indian buffet process. In *NIPS*, 2007.

[16] S. Jain and R.M. Neal. Splitting and merging components of a nonconjugate Dirichlet process mixture model (with invited discussion). *Bayesian Analysis*, 2(3):445–500, 2007.

[17] C. Wang and D. Blei. A split-merge MCMC algorithm for the hierarchical Dirichlet process. *arXiv:1201.1657v1 [stat.ML]*, 2012.

[18] E. Meeds, R. Neal, Z. Ghahramani, and S. Roweis. Modeling dyadic data with binary latent factors. In *NIPS*, 2008.

[19] K. Miller, T. Griffiths, and M. Jordan. Nonparametric latent feature models for link prediction. In *NIPS*, 2009.

[20] M. Mørup, M. N. Schmidt, and L. K. Hansen. Infinite multiple membership relational modeling for complex networks. In *IEEE International Workshop on Machine Learning for Signal Processing*, 2011.

[21] L. Tierney. Markov chains for exploring posterior distributions (with discussion). *Annals of Statistics*, 22:1701–1762, 1994.

[22] Carnegie Mellon University. Graphics lab motion capture database. http://mocap.cs.cmu.edu/.

[23] F. De la Torre et al. Guide to the Carnegie Mellon University Multimodal Activity (CMU-MMAC) database. Technical Report CMU-RI-TR-08-22, Robotics Institute, Carnegie Mellon University, 2009.

